# Differentiating Functions of the Jacobian with Respect to the Weights

**Gary William Flake**
NEC Research Institute
4 Independence Way
Princeton, NJ 08540
*flake@research.nj.nec.com*

**Barak A. Pearlmutter**
Dept of Computer Science, FEC 313
University of New Mexico
Albuquerque, NM 87131
*bap@cs.unm.edu*

## Abstract

For many problems, the correct behavior of a model depends not only on its input-output mapping but also on properties of its Jacobian matrix, the matrix of partial derivatives of the model's outputs with respect to its inputs. We introduce the *J-prop* algorithm, an efficient general method for computing the exact partial derivatives of a variety of simple functions of the Jacobian of a model with respect to its free parameters. The algorithm applies to any parametrized feedforward model, including nonlinear regression, multilayer perceptrons, and radial basis function networks.

## 1 Introduction

Let $f(x, w)$ be an $n$ input, $m$ output, twice differentiable feedforward model parameterized by an input vector, $x$, and a weight vector $w$. Its Jacobian matrix is defined as

$$J = \begin{bmatrix} \frac{\partial f_1}{\partial x_1} & \cdots & \frac{\partial f_1}{\partial x_n} \\ \vdots & \ddots & \vdots \\ \frac{\partial f_m}{\partial x_1} & \cdots & \frac{\partial f_m}{\partial x_n} \end{bmatrix} = \frac{df(x, w)}{dx}.$$

The algorithm we introduce can be used to optimize functions of the form

$$E_u(w) = \frac{1}{2} \left\| J^T u - a \right\|^2 \tag{1}$$

or

$$E_v(w) = \frac{1}{2} \left\| Jv - b \right\|^2 \tag{2}$$

where $u$, $v$, $a$, and $b$ are user-defined constants. Our algorithm, which we call *J-prop*, can be used to calculate the exact value of both $\partial E_u / \partial w$ or $\partial E_v / \partial w$ in $O(1)$ times the time required to calculate the normal gradient. Thus, J-prop is suitable for training models to have specific first derivatives, or for implementing several other well-known algorithms such as Double Backpropagation [1] and Tangent Prop [2].

Clearly, being able to optimize Equations 1 and 2 is useful; however, we suspect that the formalism which we use to derive our algorithm is actually more interesting because it allows us to modify J-prop to easily be applicable to a wide-variety of model types and

objective functions. As such, we spend a fair portion of this paper describing the mathematical framework from which we later build J-prop.

This paper is divided into four more sections. Section 2 contains background information and motivation for why optimizing the properties of the Jacobian is an important problem. Section 3 introduces our formalism and contains the derivation of the J-prop algorithm. Section 4 contains a brief numerical example of J-prop. And, finally, Section 5 describes further work and gives our conclusions.

## 2   Background and motivation

Previous work concerning the modeling of an unknown function and its derivatives can be divided into works that are descriptive or prescriptive. Perhaps the best known descriptive result is due to White *et al.* [3, 4], who show that given noise-free data, a multilayer perceptron (MLP) can approximate the higher derivatives of an unknown function in the limit as the number of training points goes to infinity. The difficulty with applying this result is the strong requirements on the amount and integrity of the training data; requirements which are rarely met in practice. This problem was specifically demonstrated by Principe, Rathie and Kuo [5] and Deco and Schürmann [6], who showed that using noisy training data from chaotic systems can lead to models that are accurate in the input-output sense, but inaccurate in their estimates of quantities related to the Jacobian of the unknown system, such as the largest Lyapunov exponent and the correlation dimension.

MLPs are particularly problematic because large weights can lead to saturation at a particular sigmoidal neuron which, in turn, results in extremely large first derivatives at the neuron when evaluated near the center of the sigmoid transition. Several methods to combat this type of over-fitting have been proposed. One of the earliest methods, weight decay [7], uses a penalty term on the magnitude of the weights. Weight decay is arguably optimal for models in which the output is linear in the weights because minimizing the magnitude of the weights is equivalent to minimizing the magnitude of the model's first derivatives. However, in the nonlinear case, weight decay can have suboptimal performance [1] because large (or small) weights do not always correspond to having large (or small) first derivatives.

The Double Backpropagation algorithm [1] adds an additional penalty term to the error function equal to $||\partial E/\partial x||^2$. Training on this function results in a form of regularization that is in many ways an elegant combination of weight decay and training with noise: it is strictly analytic (unlike training with noise) but it explicitly penalizes large first derivatives of the model (unlike weight decay). Double Backpropagation can be seen as a special case of J-prop, the algorithm derived in this paper.

As to the general problem of coercing the first derivatives of a model to specific values, Simard, *et al.*, [2] introduced the Tangent Prop algorithm, which was used to train MLPs for optical character recognition to be insensitive to small affine transformations in the character space. Tangent Prop can also be considered a special case of J-prop.

## 3   Derivation

We now define a formalism under which J-prop can be easily derived. The method is very similar to a technique introduced by Pearlmutter [8] for calculating the product of the Hessian of an MLP and an arbitrary vector. However, where Pearlmutter used differential operators applied to a model's weight space, we use differential operators defined with respect to a model's input space.

Our entire derivation is presented in five steps. First, we will define an auxiliary error

function that has a few useful mathematical properties that simplify the derivation. Next, we will define a special differential operator that can be applied to both the auxiliary error function, and its gradient with respect to the weights. We will then see that the result of applying the differential operator to the gradient of the auxiliary error function is equivalent to analytically calculating the derivatives required to optimize Equations 1 and 2. We then show an example of the technique applied to an MLP. Finally, in the last step, the complete algorithm is presented.

To avoid confusion, when referring to generic data-driven models, the model will always be expressed as a vector function $y = f(x, w)$, where $x$ refers to the model input and $w$ refers to a vector of all of the tunable parameters of the model. In this way, we can talk about models while ignoring the mechanics of how the models work internally. Complementary to the generic vector notation, the notation for an MLP uses only scalar symbols; however, these symbols must refer to internal variable of the model (e.g., neuron thresholds, net inputs, weights, etc.), which can lead to some ambiguity. To be clear, when using vector notation, the input and output of an MLP will always be denoted by $x$ and $y$, respectively, and the collection of all of the weights (including biases) map to the vector $w$. However, when using scalar arithmetic, the scalar notation for MLPs will apply.

### 3.1 Auxiliary error function

Our auxiliary error function, $\tilde{E}$, is defined as

$$\tilde{E}(x, w) = u^T f(x, w). \tag{3}$$

Note that we never actually optimize with respect $\tilde{E}$; we define it only because it has the property that $\partial \tilde{E}/\partial x = u^T J$, which will be useful to the derivation shortly. Note that $\partial \tilde{E}/\partial x$ appears in the Taylor expansion of $\tilde{E}$ about a point in input space:

$$\tilde{E}(x + \Delta x, w) = \tilde{E}(x, w) + \frac{\partial \tilde{E}}{\partial x}^T \Delta x + O\left(\|\Delta x\|^2\right). \tag{4}$$

Thus, while holding the weights, $w$, fixed and letting $\Delta x$ be a perturbation of the input, $x$, Equation 4 characterizes how small changes in the input of the model change the value of the auxiliary error function.

Be setting $\Delta x = rv$, with $v$ being an arbitrary vector and $r$ being a small value, we can rearrange Equation 4 into the form:

$$
\begin{aligned}
\frac{\partial \tilde{E}}{\partial x}^T v &= \frac{1}{r}\left[\tilde{E}(x + rv, w) - \tilde{E}(x, w)\right] + O(r) \\
&\vdots \quad = \lim_{r \to 0} \frac{1}{r}\left[\tilde{E}(x + rv, w) - \tilde{E}(x, w)\right] \\
u^T J v &= \frac{\partial}{\partial r}\tilde{E}(x + rv, w)\Big|_{r=0}.
\end{aligned}
\tag{5}
$$

This final expression will allow us to define the differential operator in the next subsection.

### 3.2 Differential operator

Let $h(x, w)$ be an arbitrary twice differentiable function. We define the differentiable operator

$$R_v\{h(x, w)\} \equiv \frac{\partial}{\partial r} h(x + rv, w)\Big|_{r=0}, \tag{6}$$

which has the property that $R_v\{\tilde{E}(x, w)\} = u^T J v$. Being a differential operator, $R_v\{\cdot\}$ obeys all of the standard rules for differentiation:

$$
\begin{aligned}
R_v\{c\} &= 0 \\
R_v\{c \cdot h(x, w)\} &= c \cdot R_v\{h(x, w)\} \\
R_v\{h(x, w) + g(x, w)\} &= R_v\{h(x, w)\} + R_v\{g(x, w)\} \\
R_v\{h(x, w) \cdot g(x, w)\} &= R_v\{h(x, w)\} \cdot g(x, w) + h(x, w) \cdot R_v\{g(x, w)\} \\
R_v\{h(g(x, w), w)\} &= h'(g(x, w)) \cdot R_v\{g(x, w)\} \\
R_v\left\{ \frac{d}{dt} h(x, w) \right\} &= \frac{d}{dt} R_v\{h(x, w)\}
\end{aligned}
$$

The operator also yields the identity $R_v\{\mathbf{x}\} = \mathbf{v}$.

### 3.3 Equivalence

We will now see that the result of calculating $R_v\{\partial \tilde{E}/\partial w\}$ can be used to calculate both $\partial E_u/\partial w$ and $\partial E_v/\partial w$. Note that Equations 3–5 all assume that both $u$ and $v$ are independent of $x$ and $w$. To calculate $\partial E_u/\partial w$ and $\partial E_v/\partial w$, we will actually set $u$ or $v$ to a value that depends on both $x$ and $w$; however, the derivation still works because our choices are explicitly made in such a way that the chain rule of differentiation is *not* supposed to be applied to these terms. Hence, the correct analytical solution is obtained despite the dependence.

To optimize with respect to Equation 1, we use:

$$
\frac{\partial}{\partial w} \frac{1}{2} \left\| J^T u - a \right\|^2 = \left( \frac{\partial u^T J}{\partial w} \right)^T (J^T u - a) = R_v\left\{ \frac{\partial \tilde{E}}{\partial w} \right\}, \tag{7}
$$

with $v = (J^T u - a)$. To optimize with respect to Equation 2, we use:

$$
\frac{\partial}{\partial w} \frac{1}{2} \left\| J v - b \right\|^2 = (J v - b)^T \left( \frac{\partial J v}{\partial w} \right) = R_v\left\{ \frac{\partial \tilde{E}}{\partial w} \right\}, \tag{8}
$$

with $u = (J v - b)$.

### 3.4 Method applied to MLPs

We are now ready to see how this technique can be applied to a specific type of model. Consider an MLP with $L + 1$ layers of nodes defined by the equations:

$$
y_i^l = g(x_i^l) \tag{9}
$$

$$
x_i^l = \sum_j^{N_l} y_j^{l-1} w_{ij}^l - \theta_i^l. \tag{10}
$$

In these equations, superscripts denote the layer number (starting at 0), subscripts index over terms in a particular layer, and $N_l$ is the number of input nodes in layer $l$. Thus, $y_i^l$ is the output of neuron $i$ at node layer $l$, and $x_i^l$ is the net input coming into the same neuron. Moreover, $y_i^L$ is an output of the entire MLP while $y_i^0$ is an input going into the MLP.

The feedback equations calculated with respect to $\tilde{E}$ are:

$$
\frac{\partial \tilde{E}}{\partial y_i^L} = u_i \tag{11}
$$

$$\frac{\partial \tilde{E}}{\partial y_i^l} = \sum_j^{N_{l+1}} w_{ij}^{l+1} \frac{\partial \tilde{E}}{\partial x_j^{l+1}} \quad \text{(for } l < L) \tag{12}$$

$$\frac{\partial \tilde{E}}{\partial x_i^l} = \frac{\partial \tilde{E}}{\partial y_i^l} g'(x_i^l) \tag{13}$$

$$\frac{\partial \tilde{E}}{\partial w_{ij}^l} = \frac{\partial \tilde{E}}{\partial x_i^l} y_j^{l-1} \tag{14}$$

$$\frac{\partial \tilde{E}}{\partial \theta_j^l} = \frac{\partial \tilde{E}}{\partial x_i^l}, \tag{15}$$

where the $u_i$ term is a component in the vector $u$ from Equation 1. Applying the $R_v\{\cdot\}$ operator to the feedforward equations yields:

$$R_v\{y_i^0\} = v_i \tag{16}$$

$$R_v\{y_i^l\} = g'(x_i^l) R_v\{x_i^l\} \quad \text{(for } l > 0) \tag{17}$$

$$R_v\{x_i^l\} = \sum_j^{N_l} R_v\{y_j^{l-1}\} w_{ij}^l, \tag{18}$$

where the $v_i$ term is a component in the vector $v$ from Equation 2. As the final step, we apply the $R_v\{\cdot\}$ operator to the feedback equations, which yields:

$$R_v\left\{\frac{\partial \tilde{E}}{\partial y_i^L}\right\} = 0 \tag{19}$$

$$R_v\left\{\frac{\partial \tilde{E}}{\partial y_i^l}\right\} = \sum_j^{N_{l+1}} w_{ij}^{l+1} R_v\left\{\frac{\partial \tilde{E}}{\partial x_j^{l+1}}\right\} \quad \text{(for } l < L) \tag{20}$$

$$R_v\left\{\frac{\partial \tilde{E}}{\partial x_i^l}\right\} = R_v\left\{\frac{\partial \tilde{E}}{\partial y_i^l}\right\} g'(x_i^l) + \frac{\partial \tilde{E}}{\partial y_i^l} g''(x_i^l) R_v\{x_i^l\} \tag{21}$$

$$R_v\left\{\frac{\partial \tilde{E}}{\partial w_{ij}^l}\right\} = R_v\left\{\frac{\partial \tilde{E}}{\partial x_i^l}\right\} y_j^{l-1} + \frac{\partial \tilde{E}}{\partial x_i^l} R_v\{y_j^{l-1}\} \tag{22}$$

$$R_v\left\{\frac{\partial \tilde{E}}{\partial \theta_j^l}\right\} = R_v\left\{\frac{\partial \tilde{E}}{\partial x_i^l}\right\}. \tag{23}$$

### 3.5 Complete algorithm

Implementing this algorithm is nearly as simple as implementing normal gradient descent. For each type of variable that is used in an MLP (net input, neuron output, weights, thresholds, partial derivatives, etc.), we require that an extra variable be allocated to hold the result of applying the $R_v\{\cdot\}$ operator to the original variable. With this change in place, the complete algorithm to compute $\partial E_u / \partial w$ is as follows:

- Set $u$ and $a$ to the user specified vectors from Equation 1.

- Set the MLP inputs to the value of $x$ that $J$ is to be evaluated at.

- Perform a normal feedforward pass using Equations 9 and 10.

- Set $\partial \tilde{E} / \partial y_i^L$ to $u_i$.

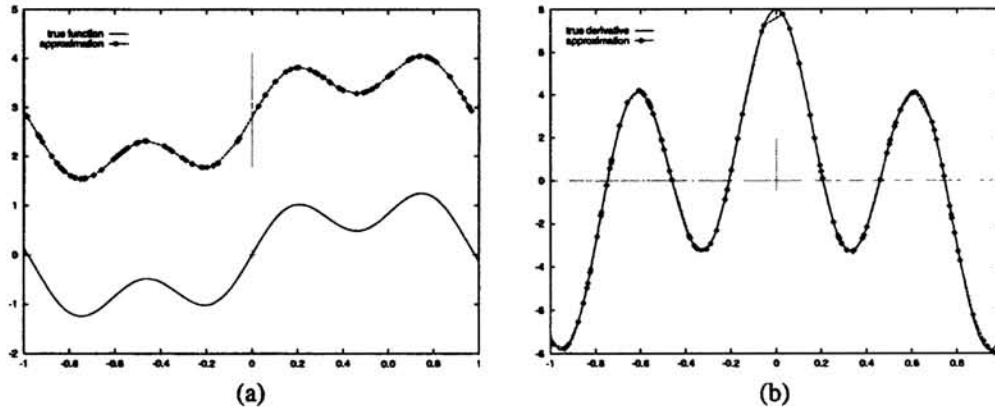

Figure 1: Learning only the derivative: showing (a) poor approximation of the function with (b) excellent approximation of the derivative.

- Perform the feedback pass with Equations 11–15. Note that values in the $\partial \tilde{E}/\partial y_i^0$ terms are now equal to $J^T u$.

- Set $v$ to $(J^T u - a)$

- Perform a $R_v\{\cdot\}$ forward pass with Equations 16–18.

- Set the $R_v\left\{\partial \tilde{E}/\partial y_i^L\right\}$ terms to 0.

- Perform a $R_v\{\cdot\}$ backward pass with Equations 19–23.

After the last step, the values in the $R_v\{\partial \tilde{E}/\partial w_{ij}^l\}$ and $R_v\{\partial \tilde{E}/\partial \theta_j^l\}$ terms contain the required result. It is important to note that the time complexity of the "J-forward" and "J-backward" calculations are nearly identical to the typical output and gradient evaluations (i.e., the "forward" and "backward" passes) of the models used.

A similar technique can be used for calculating $\partial E_v/\partial w$. The main difference is that the $R_v\{\cdot\}$ forward pass is performed between the normal forward and backward passes because $u$ can only be determined after the $R_v\{f(x, w)\}$ has been calculated.

## 4   Experimental results

To demonstrate the effectiveness and generality of the J-prop algorithm, we have implemented it on top of an existing neural network library [9] in such a way that the algorithm can be used on a large number of architectures, including MLPs, radial basis function networks, and higher order networks.

We trained an MLP with ten hidden tanh nodes on 100 points with conjugate gradient. The training exemplars consisted of inputs in $[-1, 1]$ and a target derivative from $3\cos(3x) + 5\cos(10x)$. Our unknown function (which the MLP never sees data from) is $\sin(3x) + \frac{1}{2}\sin(10x)$. The model quickly converges to a solution in approximately 100 iterations.

Figure 1 shows the performance of the MLP. Having never seen data from the unknown function, the MLP yields a poor approximation of the function, but a very accurate approximation of the function's derivative. We could have trained on both outputs and derivatives, but our goal was to illustrate that J-prop can target derivatives alone.

## 5   Conclusions

We have introduced a general method for calculating the weight gradient of functions of the Jacobian matrix of feedforward nonlinear systems. The method can be easily applied to most nonlinear models in common use today. The resulting algorithm, J-prop, can be easily modified to minimize functionals from several application domains [10]. Some possible uses include: targeting known first derivatives, implementing Tangent Prop and Double Backpropagation, enforcing identical I/O sensitivities in auto-encoders, deflating the largest eigenvalue and minimizing all eigenvalue bounds, optimizing the determinant for blind source separation, and building nonlinear controllers.

While some special cases of the J-prop algorithm have already been studied, a great deal is unknown about how optimization of the Jacobian changes the overall optimization problem. Some anecdotal evidence seems to imply that optimization of the Jacobian can lead to better generalization and faster training. It remains to be seen if J-prop used on a nonlinear extension of linear methods will lead to superior solutions.

### Acknowledgements

We thank Frans Coetzee, Yannis Kevrekidis, Joe O'Ruanaidh, Lucas Parra, Scott Rickard, Justinian Rosca, and Patrice Simard for helpful discussions. GWF would also like to thank Eric Baum and the NEC Research Institute for funding the time to write up these results.

### References

[1] H. Drucker and Y. Le Cun. Improving generalization performance using double back-propagation. *IEEE Transactions on Neural Networks*, 3(6), November 1992.

[2] P. Simard, B. Victorri, Y. Le Cun, and J. Denker. Tangent prop—A formalism for specifying selected invariances in an adaptive network. In John E. Moody, Steve J. Hanson, and Richard P. Lippmann, editors, *Advances in Neural Information Processing Systems*, volume 4, pages 895–903. Morgan Kaufmann Publishers, Inc., 1992.

[3] H. White and A. R. Gallant. On learning the derivatives of an unknown mapping with multilayer feedforward networks. In Halbert White, editor, *Artificial Neural Networks*, chapter 12, pages 206–223. Blackwell, Cambridge, Mass., 1992.

[4] H. White, K. Hornik, and M. Stinchcombe. Universal approximation of an unknown mapping and its derivative. In Halbert White, editor, *Artificial Neural Networks*, chapter 6, pages 55–77. Blackwell, Cambridge, Mass., 1992.

[5] J. Principe, A. Rathie, and J. Kuo. Prediction of chaotic time series with neural networks and the issues of dynamic modeling. *Bifurcations and Chaos*, 2(4), 1992.

[6] G. Deco and B. Schürmann. Dynamic modeling of chaotic time series. In Russell Greiner, Thomas Petsche, and Stephen José Hanson, editors, *Computational Learning Theory and Natural Learning Systems*, volume IV of *Making Learning Systems Practical*, chapter 9, pages 137–153. The MIT Press, Cambridge, Mass., 1997.

[7] G. E. Hinton. Learning distributed representations of concepts. In *Proc. Eigth Annual Conf. Cognitive Science Society*, pages 1–12, Hillsdale, NJ, 1986. Erlbaum.

[8] Barak A. Pearlmutter. Fast exact multiplication by the Hessian. *Neural Computation*, 6(1):147–160, 1994.

[9] G. W. Flake. Industrial strength modeling tools. Submitted to NIPS 99, 1999.

[10] G. W. Flake and B. A. Pearlmutter. Optimizing properties of the Jacobian of nonlinear feedforward systems. In preperation, 1999.